# Neural Network Exploration Using Optimal Experiment Design

**David A. Cohn**
Dept. of Brain and Cognitive Sciences
Massachusetts Inst. of Technology
Cambridge, MA 02139

## Abstract

Consider the problem of learning input/output mappings through exploration, e.g. learning the kinematics or dynamics of a robotic manipulator. If actions are expensive and computation is cheap, then we should explore by selecting a trajectory through the input space which gives us the most amount of information in the fewest number of steps. I discuss how results from the field of optimal experiment design may be used to guide such exploration, and demonstrate its use on a simple kinematics problem.

## 1  Introduction

Most machine learning research treats the learner as a passive receptacle for data to be processed. This approach ignores the fact that, in many situations, a learner is able, and sometimes required, to act on its environment to gather data.

Learning control inherently involves being active; the controller *must* act in order to learn the result of its action. When training a neural network to control a robotic arm, one may explore by allowing the controller to "flail" for a length of time, moving the arm at random through coordinate space while it builds up data from which to build a model [Kuperstein, 1988]. This is not feasible, however, if actions are expensive and must be conserved. In these situations, we should choose a training trajectory that will get the most information out of a limited number of steps. Manually designing such trajectories is a slow process, and intuitively "good" trajectories often fail to sufficiently explore the state space [Armstrong, 1989]. In

this paper I discuss another alternative for exploration: automatic, incremental generation of training trajectories using results from "optimal experiment design."

The study of optimal experiment design (OED) [Fedorov, 1972] is concerned with the design of experiments that are expected to minimize variances of a parameterized model. Viewing actions as experiments that move us through the state space, we can use the techniques of OED to design training trajectories.

The intent of optimal experiment design is usually to maximize confidence in a given model, minimize parameter variances for system identification, or minimize the model's output variance. Armstrong [1989] used a form of OED to identify link masses and inertial moments of a robot arm, and found that automatically generated training trajectories provided a significant improvement over human-designed trajectories. Automatic exploration strategies have been tried for neural networks (e.g. [Thrun and Möller, 1992], [Moore, 1994]), but use of OED in the neural network community has been limited. Plutowski and White [1993] successfully used it to filter a data set for maximally informative points, but its application to selecting *new* data has only been proposed [MacKay, 1992], not demonstrated.

The following section gives a brief description of the relevant results from optimal experiment design. Section 3 describes how these results may be adapted to guide neural network exploration and Section 4 presents experimental results of implementing this adaptation. Finally, Section 5 discusses implications of the results, and logical extensions of the current experiments.

## 2 Optimal experiment design

Optimal experiment design draws heavily on the technique of Maximum Likelihood Estimation (MLE) [Thisted, 1988]. Given a set of assumptions about the learner's architecture and sources of noise in the output, MLE provides a statistical basis for learning. Although the specific MLE techniques we use hold exactly only for linear models, making certain computational approximations allows them to be used with nonlinear systems such as neural networks.

We begin with a training set of input-output pairs $(x_i, y_i)_{i=1}^n$ and a learner $f_w()$. We define $f_w(x)$ to be the learner's output given input $x$ and weight vector $w$. Under an assumption of additive Gaussian noise, the maximum likelihood estimate for the weight vector, $\hat{w}$, is that which minimizes the sum squared error $E_{sse} = \sum_{i=1}^n (f_w(x_i) - y_i)^2$. The estimate $\hat{w}$ gives us an estimate of the output at a novel input: $\hat{y} = f_{\hat{w}}(x)$ (see e.g. Figure 1a).

MLE allows us to compute the variances of our weight and output estimates. Writing the output sensitivity as $g_w(x) = \partial f_w(x)/\partial w$, the covariances of $\hat{w}$ are

$$A_n^{-1} = \left(\frac{\partial^2 E_{sse}}{\partial w^2}\right)^{-1} \approx \left(\sum_{i=1}^n g_{\hat{w}}(x_i)g_{\hat{w}}(x_i)^T\right)^{-1},$$

where the last approximation assumes local linearity of $g_{\hat{w}}(x)$. (For brevity, the output sensitivity will be abbreviated to $g(x)$ in the rest of the paper.)

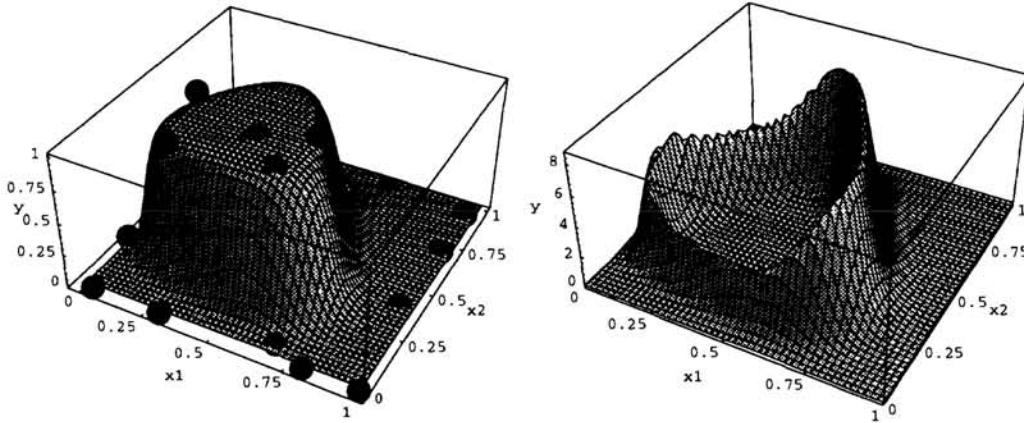

Figure 1: a) A set of training examples for a classification problem, and the network's best fit to the data. b) Maximum likelihood estimate of the network's output variance for the same problem.

For a given reference input $x_r$, the estimated output variance is

$$var(x_r) = g(x_r)^T A^{-1} g(x_r). \tag{1}$$

Output variance corresponds to the model's estimate of the expected squared distance between its output $f_{\hat{w}}(x)$ and the unknown "true" output $y$. Output variance then, corresponds to the model's estimate of its mean squared error (MSE) (see Figure 1b). If the estimates are accurate then minimizing the output variance would correspond to minimizing the network's MSE.

In optimal experiment design, we estimate how adding a new training example is expected to change the computed variances. Given a novel $x_{n+1}$, we can use OED to predict the effect of adding $x_{n+1}$ and its as-yet-unknown $y_{n+1}$ to the training set. We make the assumption that

$$A_{n+1}^{-1} \approx \left(A_n + g(x_{n+1})g(x_{n+1})^T\right)^{-1},$$

which corresponds to assuming that our current model is already fairly good. Based on this assumption, the new parameter variances will be

$$A_{n+1}^{-1} = A_n^{-1} - A_n^{-1}g(x_{n+1})(1 + g(x_{n+1})^T A_n^{-1} g(x_{n+1}))g(x_{n+1})^T A_n^{-1}.$$

Combined with Equation 1, this predicts that if we take a new example at $x_{n+1}$, the change in output variance at reference input $x_r$ will be

$$\begin{aligned} \Delta var(x_r) &= (g(x_r)^T A_n^{-1} g(x_{n+1}))^2 (1 + g(x_{n+1})^T A_n^{-1} g(x_{n+1})) \\ &= cov(x_r, x_{n+1})^2 (1 + var(x_{n+1})) \end{aligned} \tag{2}$$

To minimize the expected value of $var(x_r)$, we should select $x_{n+1}$ so as to maximize the right side of Equation 2. For other interesting OED measures, see MacKay [1992].

# 3  Adapting OED to Exploration

When building a world model, the learner is trying to build a mapping, e.g. from joint angles to cartesian coordinates (or from state-action pairs to next states). If it is allowed to select arbitrary joint angles (inputs) in successive time steps, then the problem is one of selecting the next "query" to make ([Cohn, 1990], [Baum and Lang, 1991]). In exploration, however, one's choices for a next input are constrained by the current input. We cannot instantaneously "teleport" to remote parts of the state space, but must choose among inputs that are available in the next time step.

One approach to selecting a next input is to use selective sampling: evaluate a number of possible random inputs, choose the one with the highest expected gain. In a high-dimensional action space, this is inefficient. The approach followed here is that of gradient search, differentiating Equation 2 and hillclimbing on $\partial \Delta var(x_r)/\partial x_{n+1}$.

Note that Equation 2 gives the expected change in variance only at a single point $x_r$, while we wish to minimize the average variance over the entire domain. Explicitly integrating over the domain is intractable, so we must make do with an approximation. MacKay [1992] proposed using a fixed set of reference points and measuring the expected change in variance over them. This produces spurious local maxima at the reference points, and has the undesirable effect of arbitrarily quantizing the input space. Our approach is to iteratively draw reference points at random (either uniformly or according to a distribution of interest), and compute a stochastic approximation of $\Delta var$.

By climbing the stochastically approximated gradient, either to convergence or to the horizon of available next inputs, we will settle on an input/action with a (locally) optimal decrease in expected variance.

# 4  Experimental Results

In this section, I describe two sets of experiments. The first attempts to confirm that the gains predicted by optimal experiment design may actually be realized in practice, and the second studies the application of OED to a simple learning task.

## 4.1  Expected versus actual gain

It must be emphasized that the gains predicted by OED are *expected* gains. These expectations are based on the relatively strong assumptions of MLE, which may not strictly hold. In order for the expected gains to materialize, two "bridges" must be crossed. First, the expected decrease in model variance must be realized as an actual decrease in variance. Second, the actual decrease in model variance must translate into an actual decrease in model MSE.

### 4.1.1  Expected decreases in variance → actual decreases in variance

The translation from expected to actual changes in variance requires coordination between the exploration strategy and the learning algorithm: to predict how the variance of a weight will change with a new piece of data, the predictor must know how the weight itself (and its neighboring weights) will change. Using a black

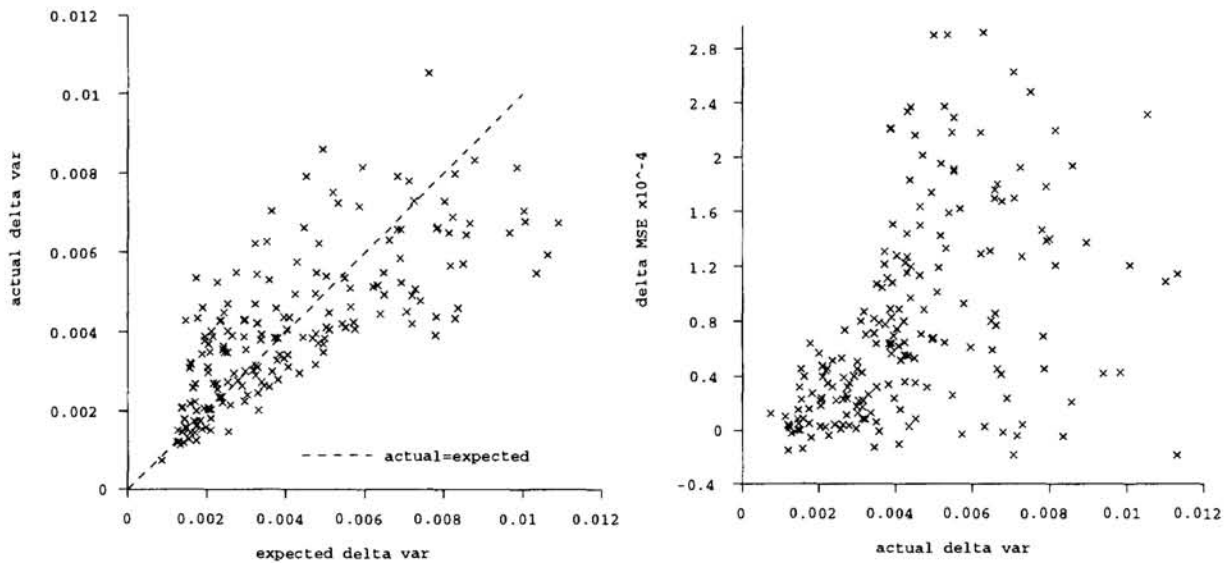

Figure 2: a) Correlations between expected change in output variance and actual change output variance b) Correlations between actual change in output variance and change in mean squared error. Correlations are plotted for a network trained on 50 examples from the arm kinematics task.

box routine like backpropagation to update the weights virtually guarantees that there will be some mismatch between expected and actual decreases in variance. Experiments indicate that, in spite of this, the correlation between predicted and actual changes in variance are relatively good (Figure 2a).

### 4.1.2   Decreases in variance $\rightarrow$ decreases in MSE

A more troubling translation is the one from model variance to model correctness. Given the highly nonlinear nature of a neural network, local minima may leave us in situations where the model is very confident but entirely wrong. Due to high confidence, the learner may reject actions that would reduce its mean squared error and explore areas where the model is correct, but has low confidence. Evidence of this behavior is seen in the lower right corner of Figure 2b, where some actions which produce a large decrease in variance have little effect on the network's MSE. While this decreases the utility of OED, it is not crippling. We discuss one possible solution to this problem at the end of this paper.

### 4.2   Learning kinematics

We have used the the stochastic approximation of $\Delta var$ to guide exploration on several simple tasks involving classification and regression. Below, I detail the experiments involving exploration of the kinematics of a simple two-dimensional, two-joint arm. The task was to learn a forward model $\Theta_1 \times \Theta_2 \rightarrow x \times y$ through exploration, which could then be used to build a controller following Jordan [1992].

The model was to be learned by a feedforward network with a sigmoid transfer function using a single hidden layer of 8 or 20 hidden units.

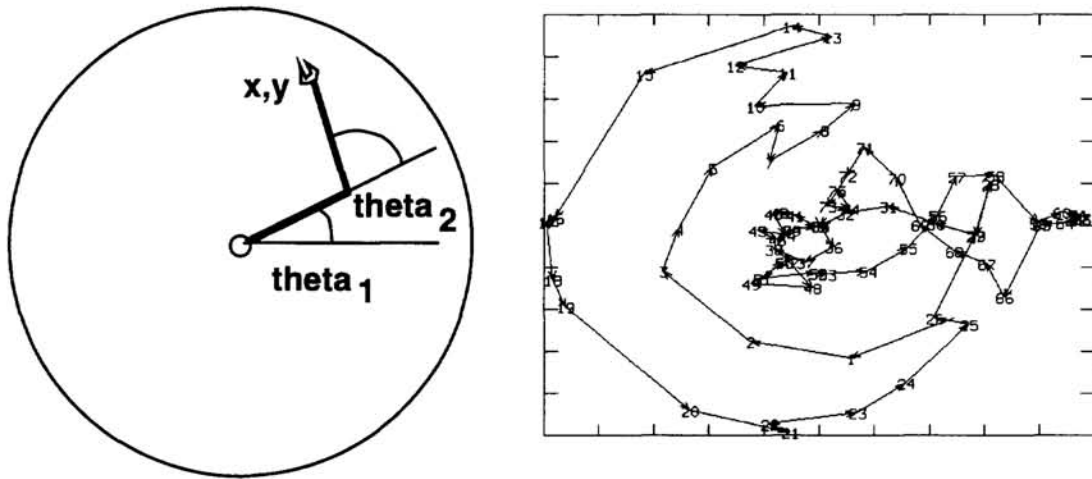

Figure 3: Learning 2D arm kinematics with 8 hidden units. a) Geometry of the 2D, two-joint arm. b) Sample trajectory using OED-based greedy exploration.

On each time step, the learner was allowed to select inputs $\Theta_1$ and $\Theta_2$ and was then given tip position $x$ and $y$ to incorporate into its training set. It then hillclimbed to find the next $\Theta_1$ and $\Theta_2$ within its limits of movement that would maximize the stochastic approximation of $\Delta var$. On each time step $\Theta_1$ and $\Theta_2$ were limited to change by no more than $\pm 36^o$ and $\pm 18^o$ respectively. Simulations were performed on the Xerion simulator (made available by the University of Toronto), approximating the variance gradient on each step with 100 randomly drawn points. A sample tip trajectory is illustrated in Figure 3b.

We compared the performance of this one-step optimal (greedy) learner, in terms of mean squared error, with that of an identical learner which explored randomly by "flailing." Not surprisingly, the improvement of greedy exploration over random exploration is significant (Figure 4b). The asymptotic performance of the greedy learner was better than that of the random learner, and it reached its asymptote in much few steps.

# 5 Discussion

The experiments described in this paper indicate that optimal experiment design is a promising tool for guiding neural network exploration. It requires no arbitrary discretization of state or action spaces, and is amenable to gradient search techniques. It does, however, have high computational costs and, as discussed in Section 4.1.2, may be led astray if the model settles in a local minimum.

## 5.1 Alternatives to greedy OED

The greedy approach is prone to "boxing itself into a corner" while leaving important parts of the domain unexplored. One heuristic for avoiding local minima is to

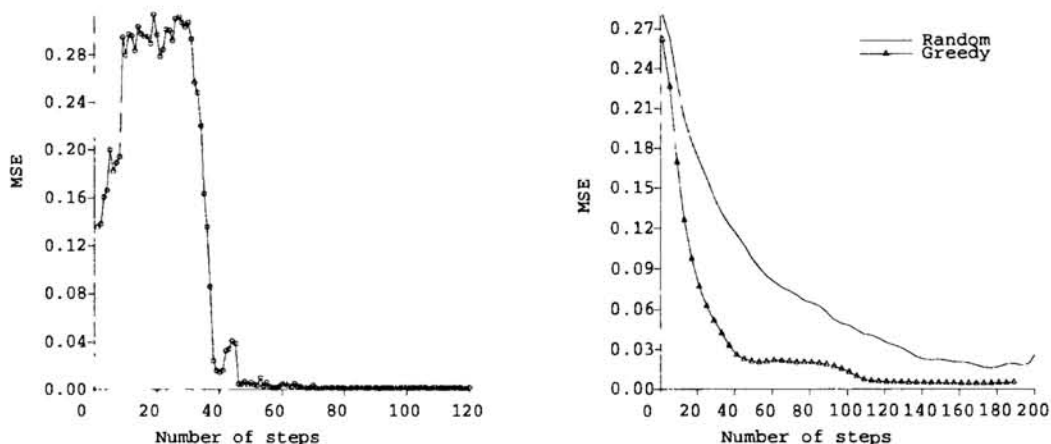

Figure 4: Learning 2D arm kinematics. a) MSE for a single exploration trajectory (20 hidden units). b) Plot of MSE for random and greedy exploration vs. number of training examples, averaged over 12 runs (8 hidden units).

occasionally check the expected gain in other parts of the input space and move towards them if they promise much greater gain than a greedy step.

The theoretically correct but computationally expensive approach is to optimize over an entire trajectory. Trajectory optimization entails starting with an initial trajectory, computing the expected gain over it, and iteratively perturbing points on the trajectory towards towards optimal expected gain (subject to other points along the trajectory being explored). Experiments are currently underway to determine how much of an improvement may be realized with trajectory optimization; it is unclear whether the improvement over the greedy approach will be worth the added computational cost.

## 5.2   Computational Costs

The computational costs of even greedy OED are great. Selecting a next action requires computation and inversion of the hessian $\partial^2 E_{sse}/\partial w^2$. Each time an action is selected and taken, the new data must be incorporated into the training set, and the learner retrained. In comparison, when using a flailing strategy or a fixed trajectory, the data may be gathered with little computation, and the learner trained only once on the batch. In this light, the cost of data must be much greater than the cost of computation for optimal experiment design to be a preferable strategy.

There are many approximations one can make which significantly bring down the cost of OED. By only considering covariances of weights leading to the same neuron, the hessian may be reduced to a block diagonal form, with each neuron computing its own (simpler) covariances in parallel. As an extreme, one can do away with covariances entirely and rely only on individual weight variances, whose computation is simple. By the same token, one can incorporate the new examples in small batches, only retraining every 5 or so steps. While suboptimal from a data gathering perspective, they appear to still outperform random exploration, and are much cheaper than "full-blown" optimization.

## 5.3   Alternative architectures

We may be able to bring down computational costs *and* improve performance by using a different architecture for the learner. With a standard feedforward neural network, not only is the repeated compution of variances expensive, it sometimes fails to yield estimates suitable for use as confidence intervals (as we saw in Section 4.1.2). A solution to both of these problems may lie in selection of a more amenable architecture and learning algorithm. One such architecture, in which output variances have a direct role in estimation, is a mixture of Gaussians, which may be efficiently trained using an EM algorithm [Ghahramani and Jordan, 1994]. We expect that it is along these lines that our future research will be most fruitful.

## Acknowledgements

I am indebted to Michael I. Jordan and David J.C. MacKay for their help in making this research possible. This work was funded by ATR Human Information Processing Laboratories, Siemens Corporate Research and NSF grant CDA-9309300.

## Bibliography

B. Armstrong. (1989) On finding exciting trajectories for identification experiments. *Int. J. of Robotics Research*, **8**(6):28–48.

E. Baum and K. Lang. (1991) Constructing hidden units using examples and queries. In R. Lippmann et al., eds., *Advances in Neural Information Processing Systems 3*, Morgan Kaufmann, San Francisco, CA.

D. Cohn, L. Atlas and R. Ladner. (1990) Training connectionist networks with queries and selective sampling. In D. Touretzky, editor, *Advances in Neural Information Processing Systems 2*, Morgan Kaufmann, San Francisco.

V. Fedorov. (1972) *Theory of Optimal Experiments*. Academic Press, New York.

Z. Ghahramani and M. Jordan. (1994) Supervised learning from incomplete data via an EM approach. *In this volume.*

M. Jordan and D. Rumelhart. (1992) Forward models: Supervised learning with a distal teacher. *Cognitive Science*, **16**(3):307–354.

D. MacKay. (1992) Information-based objective functions for active data selection, *Neural Computation* **4**(4): 590–604.

A. Moore. (1994) The parti-game algorithm for variable resolution reinforcement learning in multidimensional state-spaces. *In this volume.*

M. Plutowski and H. White. (1993) Selecting concise training sets from clean data. *IEEE Trans. on Neural Networks*, **4**(2):305–318.

R. Thisted. (1988) *Elements of Statistical Computing*. Chapman and Hall, NY.

S. Thrun and K. Möller. (1992) Active Exploration in Dynamic Environments. In J. Moody et al., editors, *Advances in Neural Information Processing Systems 4*. Morgan Kaufmann, San Francisco, CA.